# A Method for Inferring Label Sampling Mechanisms in Semi-Supervised Learning

**Saharon Rosset**
Data Analytics Research Group
IBM T.J. Watson Research Center
Yorktown Heights, NY 10598
srosset@us.ibm.com

**Ji Zhu**
Department of Statistics
University of Michigan
Ann Arbor, MI 48109
jizhu@umich.edu

**Hui Zou**
Department of Statistics
Stanford University
Stanford, CA 94305
hzou@stat.stanford.com

**Trevor Hastie**
Department of Statistics
Stanford University
Stanford, CA 94305
hastie@stanford.edu

## Abstract

We consider the situation in semi-supervised learning, where the "label sampling" mechanism stochastically depends on the true response (as well as potentially on the features). We suggest a method of moments for estimating this stochastic dependence using the unlabeled data. This is potentially useful for two distinct purposes: a. As an input to a supervised learning procedure which can be used to "de-bias" its results using labeled data only and b. As a potentially interesting learning task in itself. We present several examples to illustrate the practical usefulness of our method.

## 1  Introduction

In semi-supervised learning, we assume we have a sample $(x_i, y_i, s_i)_{i=1}^n$, of i.i.d. draws from a joint distribution on $(X, Y, S)$, where:[1]

- $x_i \in \mathbb{R}^p$ are p-vectors of features.
- $y_i$ is a label, or response ($y_i \in \mathbb{R}$ for regression, $y_i \in \{0, 1\}$ for 2-class classification).
- $s_i \in \{0, 1\}$ is a "labeling indicator", that is $y_i$ is observed if and only if $s_i = 1$, while $x_i$ is observed for all $i$.

In this paper we consider the interesting case of semi-supervised learning, where the probability of observing the response depends on the data through the true response, as well as

potentially through the features. Our goal is to model this *unknown* dependence:

$$l(x, y) = Pr(S = 1 | x, y) \tag{1}$$

Note that the dependence on $y$ (which is unobserved when $S = 0$) prevents us from using standard supervised modeling approaches to learn $l$. We show here that we can use the whole data-set (labeled+unlabeled data) to obtain estimates of this probability distribution within a parametric family of distributions, without needing to "impute" the unobserved responses.[2]

We believe this setup is of significant practical interest. Here are a couple of examples of realistic situations:

1. The problem of learning from positive examples and unlabeled data is of significant interest in document topic learning [4, 6, 8]. Consider a generalization of that problem, where we observe a sample of positive and negative examples and unlabeled data, but we believe that the positive and negative labels are supplied with different probabilities (in the document learning example, positive examples are typically more likely to be labeled than negative ones, which are much more abundant). These probabilities may also not be uniform within each class, and depend on the features as well. Our methods allow us to infer these labeling probabilities by utilizing the unlabeled data.

2. Consider a satisfaction survey, where clients of a company are requested to report their level of satisfaction, but they can choose whether or not they do so. It is reasonable to assume that their willingness to report their satisfaction depends on their actual satisfaction level. Using our methods, we can infer the dependence of the reporting probability on the actual satisfaction by utilizing the unlabeled data, i.e., the customers who declined to respond.

Being able to infer the labeling mechanism is important for two distinct reasons. First, it may be useful for "de-biasing" the results of supervised learning, which uses only the labeled examples. The generic approach for achieving this is to use "inverse sampling" weights (i.e. weigh labeled examples by $1/l(x, y)$). The us of this for maximum likelihood estimation is well established in the literature as a method for correcting sampling bias (of which semi-supervised learning is an example) [10]. We can also use the learned mechanism to post-adjust the probabilities from a probability estimation methods such as logistic regression to attain "unbiasedness" and consistency [11]. Second, understanding the labeling mechanism may be an interesting and useful learning task in itself. Consider, for example, the "satisfaction survey" scenario described above. Understanding the way in which satisfaction affects the customers' willingness to respond to the survey can be used to get a better picture of overall satisfaction and to design better future surveys, regardless of any supervised learning task which models the actual satisfaction.

Our approach is described in section 2, and is based on a method of moments. Observe that for every function of the features $g(x)$, we can get an unbiased estimate of its mean as $\frac{1}{n} \sum_{i=1}^{n} g(x_i)$. We show that if we know the underlying label sampling mechanism $l(x, y)$ we can get a different unbiased estimate of $Eg(x)$, which uses only the labeled examples, weighted by $1/l(x, y)$. We suggest inferring the *unknown* function $l(x, y)$ by requiring that we get identical estimates of $Eg(x)$ using both approaches. We illustrate our method's implementation on the California Housing data-set in section 3. In section 4 we review related work in the machine learning and statistics literature, and we conclude with a discussion in section 5.

## 2 The method

Let $g(x)$ be any function of our features. We construct two different unbiased estimates of $Eg(x)$, one based on all $n$ data points and one based on labeled examples only, assuming $P(S = 1|x, y)$ is known. Then, our method uses the equality in expectation of the two estimates to infer $P(S = 1|x, y)$. Specifically, consider $g(x)$ and also:

$$f(x, y, s) = \begin{cases} \frac{g(x)}{p(S=1|x,y)} & \text{if } s = 1 \ (y \text{ observed}) \\ 0 & \text{otherwise} \end{cases} \tag{2}$$

Then:

**Theorem 1** *Assume $P(S = 1|x, y) > 0$ , $\forall x, y$. Then:*

$$E(g(X)) = E(f(X, Y, S))$$

.

**Proof:**

$$\begin{aligned} E(f(X, Y, S)) &= \int_{X,Y,S} f(x, y, s) dP(x, y, s) = \\ &= \int_X g(x) \int_Y \frac{P(S = 1|x, y)}{P(S = 1|x, y)} dP(y|x) dP(x) = \\ &= \int_X g(x) dP(x) = Eg(X) \qquad \square \end{aligned}$$

The empirical interpretation of this expectation result is:

$$\frac{1}{n} \sum_{i=1}^{n} f(x_i, y_i, s_i) = \frac{1}{n} \sum_{i:s_i=1} \frac{g(x_i)}{P(S = 1|x_i, y_i)} \approx Eg(x) \approx \frac{1}{n} \sum_{i=1}^{n} g(x_i) \tag{3}$$

which can be interpreted as relating an estimate of $Eg(x)$ based on the complete data on the right, to the one based on labeled data only, which requires weighting that is inversely proportional to the probability of labeling, to compensate for ignoring the unlabeled data.

(3) is the fundamental result we use for our purpose, leading to a "method of moments" approach to estimating $l(x, y) = P(S = 1|x, y)$, as follows:

- Assume that $l(x, y) = p_\theta(x, y)$ , $\theta \in \mathbb{R}^k$ belongs to a parametric family with $k$ parameters.
- Select $k$ different functions $g_1(x), ..., g_k(x)$, and define $f_1, ..., f_k$ correspondingly according to (2).
- Demand equality of the leftmost and rightmost sums in (3) for each of $g_1, ..., g_k$, and solve the resulting $k$ equations to get an estimate of $\theta$.

Many practical and theoretical considerations arise when we consider what "good" choices of the representative functions $g_1(x), ..., g_k(x)$ may be. Qualitatively we would like to accomplish the standard desirable properties of inverse problems: uniqueness, stability and robustness. We want the resulting equations to have a unique "correct" solution. We want our functions to have low variance so the inaccuracy in (3) is minimal, and we want them to be "different" from each other to get a stable solution in the $k$-dimensional space. It is of course much more difficult to give concrete quantitative criteria for selecting the functions in practical situations. What we can do in practice is evaluate how stable the results we get are. We return to this topics in more detail in section 5.

A second set of considerations in selecting these functions is the computational one: can we even solve the resulting inverse problems with a reasonable computational effort? In general, solving systems of more than one nonlinear equation is a very hard problem. We also need to consider the possibility of non-unique solutions. These questions are sometimes inter-related with the choice of $g_k(x)$.

Suppose we wish to solve a set of non-linear equations for $\theta$:

$$h_k(\theta) = \sum_{s_i=1} \frac{g_k(x_i)}{p_\theta(x_i, y_i)} - \sum_i g_k(x_i) = 0, \quad k = 1, \ldots, K \tag{4}$$

The solution of (4) is similar to

$$\arg\min h(\theta) = \arg\min \sum_m h_k(\theta)^2 \tag{5}$$

Notice that every solution to (4) minimizes (5), but there may be local minima of (5) that are not solutions to (4). Hence simply applying a Newton-Raphson method to (5) is not a good idea: if we have a sufficiently good initial guess about the solution, the Newton-Raphson method converges quadratically fast; however, it can also fail to converge, if the root does not exist nearby. In practice, we can combine the Newton-Raphson method with a line search strategy that makes sure $h(\theta)$ is reduced at each iteration (the Newton step is always a descent direction of $h(\theta)$). While this method can still occasionally fail by landing on a local minimum of $h(\theta)$, this is quite rare in practice [1]. The remedy is usually to try a new starting point. Other global algorithms based on the so called *model-trust region* approach are also used in practice. These methods have a reputation for robustness even when starting far from the desired zero or minimum [2].

In some cases we can employ simpler methods, since the equations we get can be manipulated algebraically to give more "friendly" formulations. We show two examples in the next sub-section.

## 2.1 Examples of simplified calculations

We consider two situations where we can use algebra to simplify the solution of the equations our method gives. The first is the obvious application to two-class classification, where the label sampling mechanism depends on the class label only. Our method then reduces to the one suggested by [11]. The second is a more involved regression scenario, with a logistic dependence between the sampling probability and the actual label.

First, consider a two-class classification scenario, where the sampling mechanism is independent of $x$:

$$P(S = 1 | x, y) = \begin{cases} p_1 & \text{if } y = 1 \\ p_0 & \text{if } y = 0 \end{cases}$$

Then we need two functions of $x$ to "de-bias" our classes. One natural choice is $g(x) = 1$, which implies we are simply trying to invert the sampling probabilities. Assume we use one of the features $g(x) = x_j$ as our second function. Plugging these into (3) we get that to find $p_0, p_1$ we should solve:

$$n = \frac{\#\{y_i = 1 \text{ observed}\}}{\hat{p_1}} + \frac{\#\{y_i = 0 \text{ observed}\}}{\hat{p_0}}$$

$$\sum_i x_{ij} = \frac{\sum_{s_i=1, y_i=1} x_{ij}}{\hat{p_1}} + \frac{\sum_{s_i=1, y_i=0} x_{ij}}{\hat{p_0}}$$

which we can solve analytically to get:

$$\hat{p_1} = \frac{r_1 n_0 - r_0 n_1}{r n_0 - r_0 n}$$

$$\hat{p_0} = \frac{r_1 n_0 - r_0 n_1}{r_1 n - r n_1}$$

where $n_k = \#\{y_i = k \text{ observed}\}$ , $r_k = \sum_{s_i=1, y_i=k} x_{ij}$ , $k = 0, 1$

As a second, more involved, example, consider a regression situation (like the satisfaction survey mentioned in the introduction), where we assume the probability of observing the response has a linear-logistic dependence on the actual response (again we assume for simplicity independence on $x$, although dependence on $x$ poses no theoretical complications):

$$P(S = 1|x, y) = \frac{\exp(a + by)}{1 + \exp(a + by)} = \text{logit}^{-1}(a + by) \tag{6}$$

with $a, b$ unknown parameters. Using the same two $g$ functions as above gives us the slightly less friendly set of equations:

$$n = \sum_{s_i=1} \frac{1}{logit^{-1}(\hat{a} + \hat{b} y_i)}$$

$$\sum_i x_{ij} = \sum_{s_i=1} \frac{x_{ij}}{logit^{-1}(\hat{a} + \hat{b} y_i)}$$

which with some algebra we can re-write as:

$$0 = \sum_{s_i=1} \exp(-\hat{b} y_i)(\bar{x}_{0j} - x_{ij}) \tag{7}$$

$$\exp(\hat{a}) m_0 = \sum_{s_i=1} \exp(-\hat{b} y_i) \tag{8}$$

where $\bar{x}_{0j}$ is the empirical mean of the j'th feature over unlabeled examples and $m_0$ is the number of unlabeled examples. We do not have an analytic solution for these equations. However, the decomposition they offer allows us to solve them by searching first over $b$ to solve (7), then plugging the result into (8) to get an estimate of $a$. In the next section we use this solution strategy on a real-data example.

## 3 Illustration on the California Housing data-set

To illustrate our method, we take a fully labeled regression data-set and hide some of the labels based on a logistic transformation of the response, then examine the performance of our method in recovering the sampling mechanism and improving resulting prediction through de-biasing. We use the California Housing data-set [9], collected based on US Census data. It contains 20640 observations about $\log($ median house price) throughout California regions. The eight features are: median income, housing median age, total rooms, total bedrooms, population, households, latitude and longitude.

We use $3/4$ of the data for modeling and leave $1/4$ aside for evaluation. Of the training data, we hide some of the labels stochastically, based on the "label sampling" model:

$$P(S = 1|y) = \text{logit}^{-1}(1.5(y - \bar{y}) - 0.5) \tag{9}$$

this scheme results in having 6027 labeled training examples, 9372 training examples with the labels removed and 5241 test examples.

We use equations (7,8) to estimate $\hat{a}, \hat{b}$ based on each one of the 8 features. Figure 1 and Table 1 show the results of our analysis. In Figure 1 we display the value of $\sum_{s_i=1} \exp(-by_i)(\bar{x}_{0j} - x_j)$ for a range of possible values for $b$. We observe that all features give us 0 crossing around the correct value of 1.5. In Table 1 we give details of the 8 models estimated by a search strategy as follows:

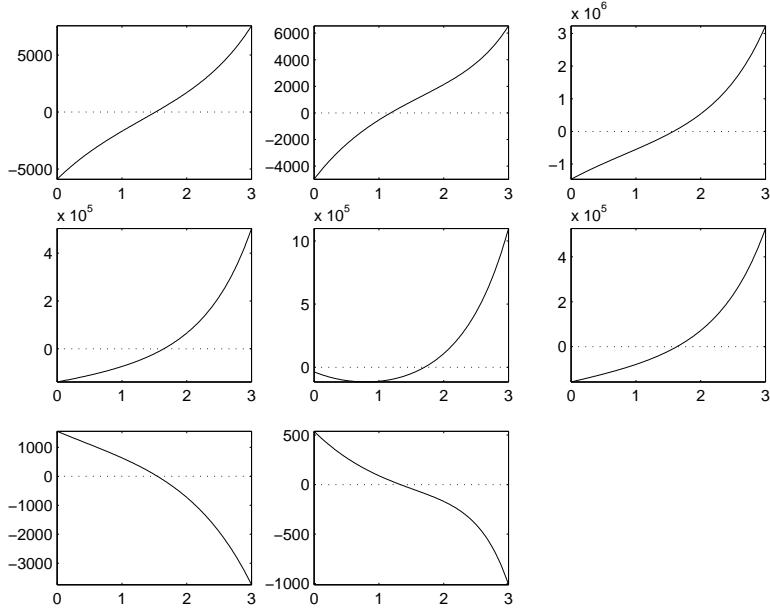

Figure 1: Value of RHS of (7) (vertical axis) vs value of $b$ (horizontal axis) for the 8 different features. The correct value is $b = 1.5$, and so we expect to observe "zero crossings" around that value, which we indeed observe on all 8 graphs.

- Find $\hat{b}$ by minimizing $|\sum_{s_i=1} \exp(-by_i)(\bar{x}_{0j} - x_{ij})|$ over the range $b \in [0, 3]$.
- Find $\hat{a}$ by plugging $\hat{b}$ from above into (8).

The table also shows the results of using these estimates to "de-bias" the prediction model, i.e. once we have $\hat{a}, \hat{b}$ we use them to calculate $\hat{P}(S = 1|y)$ and use the inverse of these estimated probabilities as weights in a least squares analysis of the labeled data. The table compares the predictive performance of the resulting models on the $1/4$ evaluation set (5241 observations) to that of the model built using labeled data only with no weighting and that of the model built using the labeled data and the "correct" weighting based on our knowledge of the true $a, b$. Most of the 8 features give reasonable estimates, and the prediction models built using the resulting weighting schemes perform comparably to the one built using the "correct" weights. They generally attain MSE about 20% smaller than that of the non-weighted model built without regard to the label sampling mechanism.

The stability of the resulting estimates is related to the "reasonableness" of the selected $g(x)$ functions. To illustrate that, we also tried the function $g(x) = x_3 \cdot x_4 \cdot x_5/(x_1 \cdot x_2)$ (still in combination with the identity function, so we can use (7,8)). The resulting estimates were $\hat{b} = 3.03$, $\hat{a} = 0.074$. Clearly these numbers are way outside the reasonable range of the results in Table 1. This is to be expected as this choice of $g(x)$ gives a function with very long tails. Thus, a few "outliers" dominate the two estimates of $E(g(x))$ in (3) which we use to estimate $a, b$.

## 4  Related work

The surge of interest in semi-supervised learning in recent years has been mainly in the context of text classification ([4, 6, 8] are several examples of many). There is also a

Table 1: Summary of estimates of sampling mechanism using each of 8 features

| Feature | b | a | Prediction MSE |
|---|---|---|---|
| median income | 1.52 | -0.519 | 0.1148 |
| housing median age | 1.18 | -0.559 | 0.1164 |
| total rooms | 1.58 | -0.508 | 0.1147 |
| total bedrooms | 1.64 | -0.497 | 0.1146 |
| population | 1.7 | -0.484 | 0.1146 |
| households | 1.63 | -0.499 | 0.1146 |
| latitude | 1.55 | -0.514 | 0.1147 |
| longitude | 1.33 | -0.545 | 0.1155 |
| (*no weighting*) | N/A | N/A | 0.1354 |
| (*true sampling model*) | 1.5 | -0.5 | 0.1148 |

wealth of statistical literature on missing data and biased sampling (e.g. [3, 7, 10]) where methods have been developed that can be directly or indirectly applied to semi-supervised learning. Here we briefly survey some of the interesting and popular approaches and relate them to our method.

The EM approach to text classification is advocated by [8]. Some ad-hoc two-step variants are surveyed by [6]. They consists of iterating between completing class labels and estimating the classification model. The main caveat of all these methods is that they ignore the sampling mechanism, and thus implicitly assume it cancels out in the likelihood function — i.e., that the sampling is random and that $l(x, y)$ is fixed. It is possible, in principle, to remove this assumption, but that would significantly increase the complexity of the algorithms, as it would require specifying a likelihood model for the sampling mechanism and adding its parameters to the estimation procedure. The methods described by [7] and discussed below take this approach.

The use of re-weighted loss to account for unknown sampling mechanisms is suggested by [4, 11]. Although they differ significantly in the details, both of these can account for label-dependent sampling in two-class classification. They do not offer solutions for other modeling tasks or for feature-dependent sampling, which our approach covers.

In the missing data literature, [7] (chapter 15) and references therein offer several methods for handling "nonignorable nonresponse". These are all based on assuming complete probability models for $(X, Y, S)$ and designing EM algorithms for the resulting problem. An interesting example is the *bivariate normal stochastic ensemble* model, originally suggested by [3]. In our notation, they assume that there is an additional fully unobserved "response" $z_i$, and that $y_i$ is observed if and only if $z_i > 0$. They also assume that $y_i, z_i$ are bivariate normal, depending on the features $x_i$, that is:

$$\begin{pmatrix} y_i \\ z_i \end{pmatrix} \sim N \left[ \begin{pmatrix} x_i\beta_1 \\ x_i\beta_2 \end{pmatrix}, \begin{pmatrix} \sigma^2 & \rho\sigma^2 \\ \rho\sigma^2 & 1 \end{pmatrix} \right]$$

this leads to a complex, but manageable, EM algorithm for inferring the sampling mechanism and fitting the actual model at once. The main shortcoming of this approach, as we see it, is in the need to specify a complete and realistic joint probability model engulfing both the sampling mechanism and the response function. This precludes completely the use of non-probabilistic methods for the response model part (like trees or kernel methods), and seems to involve significant computational complications if we stray from normal distributions.

# 5 Discussion

The method we suggest in this paper allows for the separate and unbiased estimation of label-sampling mechanisms, even when they stochastically depend on the partially unobserved labels. We view this "de-coupling" of the sampling mechanism estimation from the actual modeling task at hand as an important and potentially very useful tool, both because it creates a new, interesting learning task and because the results of the sampling model can be used to "de-bias" *any* black-box modeling tool for the supervised learning task through inverse weighting (or sampling, if the chosen tool does not take weights).

Our method of moments suffers from the same problems all such methods (and inverse problems in general) share, namely the uncertainty about the stability and validity of the results. It is difficult to develop general theory for stable solutions to inverse problems. What we can do in practice is attempt to validate the estimates we get. We have already seen one approach for doing this in section 3, where we used multiple choices for $g(x)$ and compared the resulting estimates of the parameters determining $l(x, y)$. Even if we had not known the true values of $a$ and $b$, the fact that we got similar estimates using different features would reassure us that these estimates were reliable, and give us an idea of their uncertainty. A second approach for evaluating the resulting estimates could be to use bootstrap sampling, which can be used to calculate bootstrap confidence intervals of the parameter estimates.

The computational issues also need to be tackled if our method is to be applicable for large scale problems with complex sampling mechanisms. We have suggested a general methodology in section 2, and some ad-hoc solutions for special cases in section 2.1. However we feel that a lot more can be done to develop efficient and widely applicable methods for solving the moment equations.

### Acknowledgments

We thank John Langford and Tong Zhang for useful discussions.

## Footnotes

[1]Our notation here differs somewhat from many semi-supervised learning papers, where the unlabeled part of the sample is separated from the labeled part and sometimes called "test set".

[2]The importance of this is that we are required to hypothesize and fit a *conditional* probability model for $l(x, y)$ only, as opposed to the full probability model for $(S, X, Y)$ required for, say, EM.

# References

[1]   Acton, F. (1990) *Numerical Methods That Work*. Washington: Math. Assoc. of America.

[2]   Dennis, J. & Schnabel, R. (1983) *Numerical Methods for Unconstrained Optimization and Nonlinear Equations*. New Jersey: Prentice-Hall.

[3]   Heckman, J.I. (1976). The common structure of statistical models for truncation, sample selection and limited dependent variables, and a simple estimator for such models. *Annals of Economic and Social Measurement* 5:475-492.

[4]   Lee, W.S. & Liu, B. (2003). Learning with Positive and Unlabeled Examples Using Weighted Logistic Regression. *ICML-03*

[5]   Lin, Y., Lee, Y. & Wahba, G. (2000). Support vector machines for classification in nonstandard situations. *Machine Learning*, 46:191-202.

[6]   Liu, B., Dai, Y., Li, X., Lee, W.S. & Yu, P. (2003). Building Text Classifiers Using Positive and Unlabeled Examples. *Proceedings ICDM-03*

[7]   Little, R. & Rubin, D. (2002). *Statistical Analysis with Missing Data, 2nd Ed.* . Wiley & Sons.

[8]   Nigam, K., McCallum , A., Thrun, S. & Mitchell, T. (2000) Text Classification from Labeled and Unlabeled Documents using EM. *Machine Learning* 39(2/3):103-134.

[9]   Pace, R.K. & Barry, R. (1997). Sparse Spatial Autoregressions. *Stat. & Prob. Let.*, 33 291-297.

[10]  Vardi, Y. (1985). Empirical Distributions in Selection Bias Models. *Annals of Statistics*, 13.

[11]  Zou, H., Zhu, J. & Hastie, T. (2004). Automatic Bayes Carpentary in Semi-Supervised Classification. *Unpublished.*
